# Nonlinear processing in LGN neurons

**Vincent Bonin**[*], **Valerio Mante and Matteo Carandini**

Smith-Kettlewell Eye Research Institute
2318 Fillmore Street
San Francisco, CA 94115, USA

Institute of Neuroinformatics
University of Zurich and ETH Zurich
Winterthurerstrasse 190
CH-8046 Zurich, Switzerland

*{vincent,valerio,matteo}@ski.org*

## Abstract

According to a widely held view, neurons in lateral geniculate nucleus (LGN) operate on visual stimuli in a linear fashion. There is ample evidence, however, that LGN responses are not entirely linear. To account for nonlinearities we propose a model that synthesizes more than 30 years of research in the field. Model neurons have a linear *receptive field*, and a nonlinear, divisive *suppressive field*. The suppressive field computes local root-mean-square contrast. To test this model we recorded responses from LGN of anesthetized paralyzed cats. We estimate model parameters from a basic set of measurements and show that the model can accurately predict responses to novel stimuli. The model might serve as the new standard model of LGN responses. It specifies how visual processing in LGN involves both linear filtering and divisive gain control.

## 1 Introduction

According to a widely held view, neurons in lateral geniculate nucleus (LGN) operate linearly (Cai et al., 1997; Dan et al., 1996). Their response $L(t)$ is the convolution of the map of stimulus contrast $S(\mathbf{x},t)$ with a receptive field $F(\mathbf{x},t)$:

$$L(t) = \left[ S * F \right]\left( \mathbf{0}, t \right)$$

The receptive field $F(\mathbf{x},t)$ is typically taken to be a difference of Gaussians in space (Rodieck, 1965) and a difference of Gamma functions in time (Cai et al., 1997).

This linear model accurately predicts the selectivity of responses for spatiotemporal frequency as measured with gratings (Cai et al., 1997; Enroth-Cugell and Robson, 1966). It also predicts the main features of responses to complex dynamic video sequences (Dan et al., 1996).

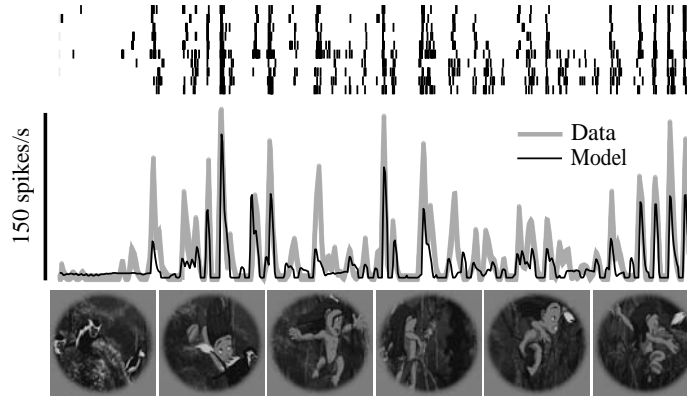

Figure 1. Response of an LGN neuron to a dynamic video sequence along with the prediction made by the linear model. Stimuli were sequences from Walt Disney's "Tarzan". From Mante et al. (2002).

The linear model, however, suffers from limitations. For example, consider the response of an LGN neuron to a complex dynamic video sequences (Figure 1). The response is characterized by long periods of relative silence interspersed with brief events of high firing rate (Figure 1, *thick traces*). The linear model (Figure 1, *thin traces*) successfully predicts the timing of these firing events but fails to account for their magnitude (Mante et al., 2002).

The limitations of the linear model are not surprising since there is ample evidence that LGN responses are nonlinear. For instance, responses to drifting gratings saturate as contrast is increased (Sclar et al., 1990) and are reduced, or *masked*, by superposition of a second grating (Bonin et al., 2002). Moreover, responses are selective for stimulus size (Cleland et al., 1983; Hubel and Wiesel, 1961; Jones and Sillito, 1991) in a nonlinear manner (Solomon et al., 2002).

We propose that these and other nonlinearities can be explained by a nonlinear model incorporating a nonlinear *suppressive field*. The qualitative notion of a suppressive field was proposed three decades ago by Levick and collaborators (1972). We propose that the suppressive field computes local root-mean-square contrast, and operates divisively on the receptive field output.

Basic elements of this model appeared in studies of contrast gain control in retina (Shapley and Victor, 1978) and in primary visual cortex (Cavanaugh et al., 2002; Heeger, 1992; Schwartz and Simoncelli, 2001). Some of these notions have been applied to LGN (Solomon et al., 2002), to fit responses to a limited set of stimuli with tailored parameter sets. Here we show that a single model with fixed parameters predicts responses to a broad range of stimuli.

## 2 Model

In the model (Figure 2), the linear response of the receptive field $L(t)$ is divided by the output of the suppressive field. The latter is a measure of local root-mean-square contrast $c_{local}$. The result of the division is a generator potential

$$V(t) = V_{max} \frac{L(t)}{c_{50} + c_{local}},$$

where $c_{50}$ is a constant.

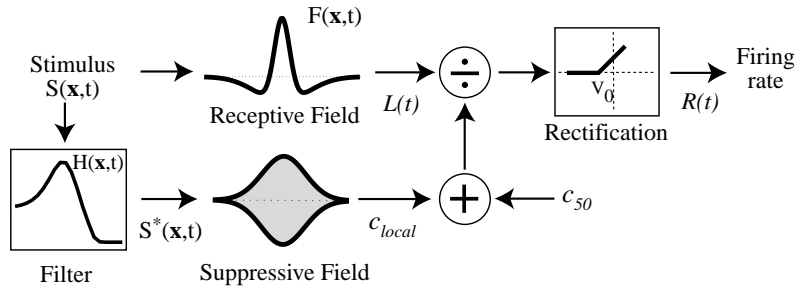

Figure 2. Nonlinear model of LGN responses.

The suppressive field operates on a filtered version of the stimulus, $S^*=S*H$, where H is a linear filter and $*$ denotes convolution. The squared output of the suppressive field is the local mean square (the local variance) of the filtered stimulus:

$$c_{\text{local}}^2 = \iint S^* (\mathbf{x},t)^2 \, G(\mathbf{x}) \, d\mathbf{x} \, dt ,$$

where $G(\mathbf{x})$ is a 2-dimensional Gaussian.

Firing rate is a rectified version of generator potential, with threshold $V_{\text{thresh}}$:

$$R(t) = \lfloor V(t) - V_{thresh} \rfloor_+ .$$

To test the nonlinear model, we recorded responses from neurons in the LGN of anesthetized paralyzed cats. Methods for these recordings were described elsewhere (Freeman et al., 2002).

## 3   Results

We proceed in two steps: first we estimate model parameters by fitting the model to a large set of canonical data; second we fix model parameters and evaluate the model by predicting responses to a novel set of stimuli.

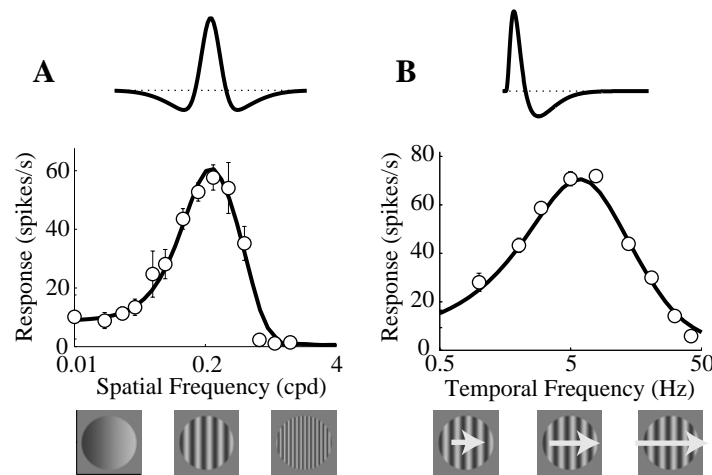

Figure 3. Estimating the receptive field in an example LGN cell. Stimuli are gratings varying in spatial (**A**) and temporal (**B**) frequency. Responses are the harmonic component of spike trains at the grating temporal frequency. Error bars represent standard deviation of responses. Curves indicate model fit.

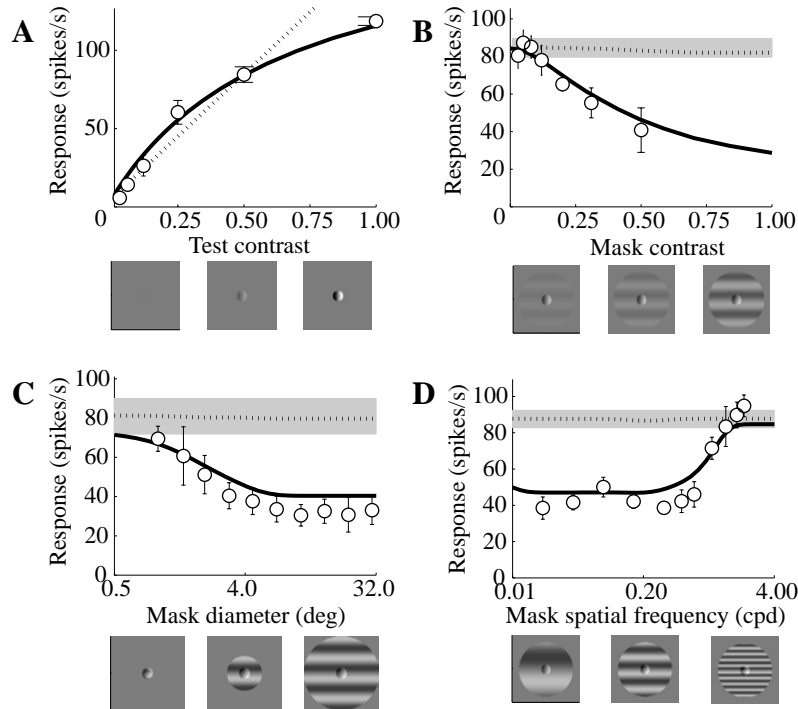

Figure 4. Estimating the suppressive field in the example LGN cell. Stimuli are sums of a *test* grating and a *mask* grating. Responses are the harmonic component of spike trains at the temporal frequency of test. **A**: Responses to test alone. **B-D**: Responses to test+mask as function of three mask attributes: contrast (**B**), diameter (**C**) and spatial frequency (**D**). Gray areas indicate baseline response (test alone, 50% contrast). Dashed curves are predictions of linear model. Solid curves indicate fit of nonlinear model.

## 3.1 Characterizing the receptive field

We obtain the parameters of the receptive field $F(\mathbf{x},t)$ from responses to large drifting gratings (Figure 3). These stimuli elicit approximately constant output in the suppressive field, so they allow us to characterize the receptive field. Responses to different spatial frequencies constrain $F(\mathbf{x},t)$ in space (Figure 3**A**). Responses to different temporal frequencies constrain $F(\mathbf{x},t)$ in time (Figure 3**B**).

## 3.2 Characterizing the suppressive field

To characterize the divisive stage, we start by measuring how responses saturate at high contrast (Figure 4**A**). A linear model cannot account for this contrast saturation (Figure 4**A**, *dashed curve*). The nonlinear model (Figure 4**A**, *solid curve*) captures saturation because increases in receptive field output are attenuated by increases in suppressive field output. At low contrast, no saturation is observed because the output of the suppressive field is dominated by the constant $c_{50}$. From these data we estimate the value of $c_{50}$.

To obtain the parameters of the suppressive field, we recorded responses to sums of two drifting gratings (Figure 4**B**-**D**): an optimal *test* grating at 50% contrast, which elicits a large baseline response, and a *mask* grating that modulates this response. Test and mask temporal frequencies are incommensurate so that they temporally label a test response (at the frequency of the test) and a mask response (at the

frequency of the mask) (Bonds, 1989). We vary mask attributes and study how they affect the test responses.

Increasing mask contrast progressively suppresses responses (Figure 4**B**). The linear model fails to account for this suppression (Figure 4**B**, *dashed curve*). The nonlinear model (Figure 4**B**, *solid curve*) captures it because increasing mask contrast increases the suppressive field output while the receptive field output (at the temporal frequency of the test) remains constant. With masks of low contrast there is little suppression because the output of the suppressive field is dominated by the constant $c_{50}$.

Similar effects are seen if we increase mask diameter. Responses decrease until they reach a plateau (Figure 4**C**). A linear model predicts no decrease (Figure 4**C**, *dashed curve*). The nonlinear model (Figure 4**C**, *solid curve*) captures it because increasing mask diameter increases the suppressive field output while it does not affect the receptive field output. A plateau is reached once masks extend beyond the suppressive field. From these data we estimate the size of the Gaussian envelope $G(\mathbf{x})$ of the suppressive field.

Finally, the strength of suppression depends on mask spatial frequency (Figure 4**D**). At high frequencies, no suppression is elicited. Reducing spatial frequency increases suppression. This dependence of suppression on spatial frequency is captured in the nonlinear model by the filter $H(\mathbf{x},t)$. From these data we estimate the spatial characteristics of the filter. From similar experiments involving different temporal frequencies (not shown), we estimate the filter's selectivity for temporal frequency.

## 3.3   Predicting responses to novel stimuli

We have seen that with a fixed set of parameters the model provides a good fit to a large set of measurements (Figure 3 and Figure 4). We now test whether the model predicts responses to a set of novel stimuli: drifting gratings varying in contrast and diameter.

Responses to high contrast stimuli exhibit size tuning (Figure 5**A**, *squares*): they grow with size for small diameters, reach a maximum value at intermediate diameter and are reduced for large diameters (Jones and Sillito, 1991). Size tuning , however, strongly depends on stimulus contrast (Solomon et al., 2002): no size tuning is observed at low contrast (Figure 5**A**, *circles*). The model predicts these effects (Figure 5**A**, *curves*). For large, high contrast stimuli the output of the suppressive field is dominated by $c_{local}$, resulting in suppression of responses. At low contrast, $c_{local}$ is much smaller than $c_{50}$, and the suppressive field does not affect responses.

Similar considerations can be made by plotting these data as a function of contrast (Figure 5**B**). As predicted by the nonlinear model (Figure 5**B**, *curves*), the effect of increasing contrast depends on stimulus size: responses to large stimuli show strong saturation (Figure 5**B**, *squares*), whereas responses to small stimuli grow linearly (Figure 5**B**, *circles*). The model predicts these effects because only large, high contrast stimuli elicit large enough responses from the suppressive field to cause suppression. For small, low contrast stimuli, instead, the linear model is a good approximation.

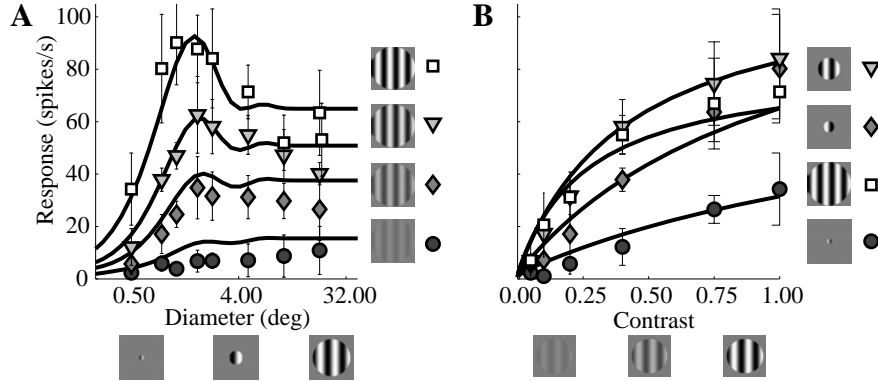

Figure 5. Predicting responses to novel stimuli in the example LGN cell. Stimuli are gratings varying in diameter and contrast, and responses are harmonic component of spike trains at grating temporal frequency. Curves show model predictions based on parameters as estimated in previous figures, not fitted to these data. **A**: Responses as function of diameter for different contrasts. **B**: Responses as function of contrast for different diameters.

## 3.4 Model performance

To assess model performance across neurons we calculate the percentage of variance in the data that is explained by the model (see Freeman et al., 2002 for methods).

The model provides good fits to the data used to characterize the suppressive field (Figure 4), explaining more than 90% of the variance in the data for 9/13 cells (Figure 6**A**). Model parameters are then held fixed, and the model is used to predict responses to gratings of different contrast and diameter (Figure 5). The model performs well, explaining in 10/13 neurons above 90% of the variance in these novel data (Figure 6**B**, *shaded histogram*). The agreement between the quality of the fits and the quality of the predictions suggests that model parameters are well constrained and rules out a role of overfitting in determining the quality of the fits.

To further confirm the performance of the model, in an additional 54 cells we ran a subset of the whole protocol, involving only the experiment for characterizing the receptive field (Figure 3), and the experiment involving gratings of different contrast and diameter (Figure 5). For these cells we estimate the suppressive field by fitting the model directly to the latter measurements. The model explains above 90% of the variance in these data in 20/54 neurons and more than 70% in 39/54 neurons (Figure 6**B**, *white histogram*).

Considering the large size of the data set (more than 100 stimuli, requiring several hours of recordings per neuron) and the small number of free parameters (only 6 for the purpose of this work), the overall, quality of the model predictions is remarkable.

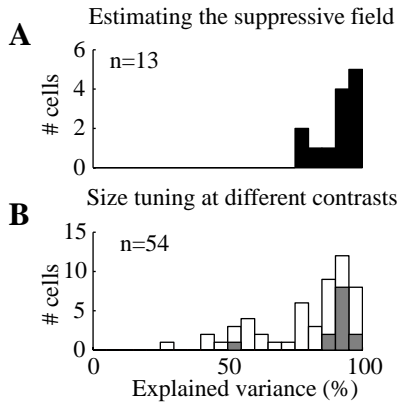

Figure 6. Percentage of variance in data explained by model. **A**: Experiments to estimate the suppressive field. **B**: Experiments to test the model. Gray histogram shows quality of predictions. White histogram shows quality of fits.

## 4 Conclusions

The nonlinear model provides a unified description of visual processing in LGN neurons. Based on a fixed set of parameters, it can predict both linear properties (Figure 3), as well as nonlinear properties such as contrast saturation (Figure 4**A**) and masking (Figure 4**B-D**). Moreover, once the parameters are fixed, it predicts responses to novel stimuli (Figure 5).

The model explains why responses are tuned for stimulus size at high contrast but not at low contrast, and it correctly predicts that only responses to large stimuli saturate with contrast, while responses to small stimuli grow linearly.

The model implements a form of contrast gain control. A possible purpose for this gain control is to increase the range of contrast that can be transmitted given the limited dynamic range of single neurons. Divisive gain control may also play a role in population coding: a similar model applied to responses of primary visual cortex was shown to maximize independence of the responses across neurons (Schwartz and Simoncelli, 2001).

We are working towards improving the model in two ways. First, we are characterizing the dynamics of the suppressive field, e.g. to predict how it responds to transient stimuli. Second, we are testing the assumption that the suppressive field computes root-mean-square contrast, a measure that solely depends on the second-order moments of the light distribution.

Our ultimate goal is to predict responses to complex stimuli such as those shown in Figure 1 and quantify to what degree the nonlinear model improves on the predictions of the linear model. Determining the role of visual nonlinearities under more natural stimulation conditions is also critical to understanding their function.

The nonlinear model synthesizes more than 30 years of research. It is robust, tractable and generalizes to arbitrary stimuli. As a result it might serve as the new standard model of LGN responses. Because the nonlinearities we discussed are already present in the retina (Shapley and Victor, 1978), and tend to get stronger as one ascends the visual hierarchy (Sclar et al., 1990), it may also be used to study how responses take shape from one stage to another in the visual system.

## Acknowledgments

This work was supported by the Swiss National Science Foundation and by the James S McDonnell Foundation 21st Century Research Award in Bridging Brain, Mind & Behavior.

## References

Bonds, A. B. (1989). Role of inhibition in the specification of orientation selectivity of cells in the cat striate cortex. Vis Neurosci *2*, 41-55.

Bonin, V., Mante, V., and Carandini, M. (2002). The contrast integration field of cat LGN neurons. Program No. 352.16. In Abstract Viewer/Itinerary Planner (Washington, DC, Society for Neuroscience).

Cai, D., DeAngelis, G. C., and Freeman, R. D. (1997). Spatiotemporal receptive field organization in the lateral geniculate nucleus of cats and kittens. J Neurophysiol *78*, 1045-1061.

Cavanaugh, J. R., Bair, W., and Movshon, J. A. (2002). Selectivity and spatial distribution of signals from the receptive field surround in macaque v1 neurons. J Neurophysiol *88*, 2547-2556.

Cleland, B. G., Lee, B. B., and Vidyasagar, T. R. (1983). Response of neurons in the cat's lateral geniculate nucleus to moving bars of different length. J Neurosci *3*, 108-116.

Dan, Y., Atick, J. J., and Reid, R. C. (1996). Efficient coding of natural scenes in the lateral geniculate nucleus: experimental test of a computational theory. J Neurosci *16*, 3351-3362.

Enroth-Cugell, C., and Robson, J. G. (1966). The contrast sensitivity of retinal ganglion cells of the cat. J Physiol (Lond) *187*, 517-552.

Freeman, T., Durand, S., Kiper, D., and Carandini, M. (2002). Suppression without Inhibition in Visual Cortex. Neuron *35*, 759.

Heeger, D. J. (1992). Normalization of cell responses in cat striate cortex. Vis Neurosci *9*, 181-197.

Hubel, D., and Wiesel, T. N. (1961). Integrative action in the cat's lateral geniculate body. J Physiol (Lond) *155*, 385-398.

Jones, H. E., and Sillito, A. M. (1991). The length-response properties of cells in the feline dorsal lateral geniculate nucleus. J Physiol (Lond) *444*, 329-348.

Levick, W. R., Cleland, B. G., and Dubin, M. W. (1972). Lateral geniculate neurons of cat: retinal inputs and physiology. Invest Ophthalmol *11*, 302-311.

Mante, V., Bonin, V., and Carandini, M. (2002). Responses of cat LGN neurons to plaids and movies. Program No. 352.15. In Abstract Viewer/Itinerary Planner (Washington, DC, Society for Neuroscience).

Rodieck, R. W. (1965). Quantitative analysis of cat retina ganglion cell response to visual stimuli. Vision Res *5*, 583-601.

Schwartz, O., and Simoncelli, E. P. (2001). Natural signal statistics and sensory gain control. Nat Neurosci *4*, 819-825.

Sclar, G., Maunsell, J. H. R., and Lennie, P. (1990). Coding of image contrast in central visual pathways of the macaque monkey. Vision Res *30*, 1-10.

Shapley, R. M., and Victor, J. D. (1978). The effect of contrast on the transfer properties of cat retinal ganglion cells. J Physiol *285*, 275-298.

Solomon, S. G., White, A. J., and Martin, P. R. (2002). Extraclassical receptive field properties of parvocellular, magnocellular, and koniocellular cells in the primate lateral geniculate nucleus. J Neurosci *22*, 338-349.
